# A reinterpretation of the policy oscillation phenomenon in approximate policy iteration

**Paul Wagner**
Department of Information and Computer Science
Aalto University School of Science
PO Box 15400, FI-00076 Aalto, Finland
`pwagner@cis.hut.fi`

## Abstract

A majority of approximate dynamic programming approaches to the reinforcement learning problem can be categorized into greedy value function methods and value-based policy gradient methods. The former approach, although fast, is well known to be susceptible to the policy oscillation phenomenon. We take a fresh view to this phenomenon by casting a considerable subset of the former approach as a limiting special case of the latter. We explain the phenomenon in terms of this view and illustrate the underlying mechanism with artificial examples. We also use it to derive the constrained natural actor-critic algorithm that can interpolate between the aforementioned approaches. In addition, it has been suggested in the literature that the oscillation phenomenon might be subtly connected to the grossly suboptimal performance in the Tetris benchmark problem of all attempted approximate dynamic programming methods. We report empirical evidence against such a connection and in favor of an alternative explanation. Finally, we report scores in the Tetris problem that improve on existing dynamic programming based results.

## 1   Introduction

We consider the reinforcement learning problem in which one attempts to find a good policy for controlling a stochastic nonlinear dynamical system. Many approaches to the problem are value-based and build on the methodology of simulation-based approximate dynamic programming [1, 2, 3, 4, 5]. In this setting, there is no fixed set of data to learn from, but instead the target system, or typically a simulation of it, is actively sampled during the learning process. This learning setting is often described as interactive learning (e.g., [1, §3]).

The majority of these methods can be categorized into greedy value function methods (critic-only) and value-based policy gradient methods (actor-critic) (e.g., [1, 6]). The former approach, although fast, is susceptible to potentially severe policy oscillations in presence of approximations. This phenomenon is known as the policy oscillation (or policy chattering) phenomenon [7, 8]. The latter approach has better convergence guarantees, with the strongest case being for Monte Carlo evaluation with 'compatible' value function approximation. In this case, convergence w.p.1 to a local optimum can be established under mild assumptions [9, 6, 4].

Bertsekas has recently called attention to the currently not well understood policy oscillation phenomenon [7]. He suggests that a better understanding of it is needed and that such understanding "has the potential to alter in fundamental ways our thinking about approximate DP." He also notes that little progress has been made on this topic in the past decade. In this paper, we will try to shed more light on this topic. The motivation is twofold. First, the policy oscillation phenomenon is intimately connected to some aspects of the learning dynamics at the very heart of approximate dynamic

---

An extended version of this paper is available at `http://users.ics.tkk.fi/pwagner/`.

programming; the lack of understanding in the former implies a lack of understanding in the latter. In the long run, this state might well be holding back important theoretical developments in the field. Second, methods not susceptible to oscillations have a much better suboptimality bound [7], which gives also immediate value to a better understanding of oscillation-predisposing conditions.

The policy oscillation phenomenon is strongly associated in the literature with the popular Tetris benchmark problem. This problem has been used in numerous studies to evaluate different learning algorithms (see [10, 11]). Several studies, including [12, 13, 14, 11, 15, 16, 17], have been conducted using a standard set of features that were originally proposed in [12]. This setting has posed considerable difficulties to some approximate dynamic programming methods. Impressively fast initial improvement followed by severe degradation was reported in [12] using a greedy approximate policy iteration method. This degradation has been taken in the literature as a manifestation of the policy oscillation phenomenon [12, 8].

Policy gradient and greedy approximate value iteration methods have shown much more stable behavior in the Tetris problem [13, 14], although it has seemed that this stability tends to come at the price of speed (see esp. [13]). Still, the performance levels reached by even these methods fall way short of what is known to be possible. The typical performance levels obtained with approximate dynamic programming methods have been around 5,000 points [12, 8, 13, 16], while an improvement to around 20,000 points has been obtained in [14] by considerably *lowering* the discount factor. On the other hand, performance levels between 300,000 and 900,000 points were obtained recently with the very same features using the cross-entropy method [11, 15]. It has been hypothesized in [7] that this grossly suboptimal performance of even the best-performing approximate dynamic programming methods might also have some subtle connection to the oscillation phenomenon. In this paper, we will also briefly look into these potential connections.

The structure of the paper is as follows. After providing background in Section 2, we discuss the policy oscillation phenomenon in Section 3 along with three examples, one of which is novel and generalizes the others. We develop a novel view to the policy oscillation phenomenon in Sections 4 and 5. We validate the view also empirically in Section 6 and proceed to looking for the suggested connection between the oscillation phenomenon and the convergence issues in the Tetris problem. We report empirical evidence that indeed suggests a shared explanation to the policy degradation observed in [12, 8] and the early stagnation of all the rest of the attempted approximate dynamic programming methods. However, it seems that this explanation is not primarily related to the oscillation phenomenon but to numerical instability.

## 2   Background

A Markov decision process (MDP) is defined by a tuple $\mathcal{M} = (\mathcal{S}, \mathcal{A}, \mathcal{P}, r)$, where $\mathcal{S}$ and $\mathcal{A}$ denote the state and action spaces. $S_t \in \mathcal{S}$ and $A_t \in \mathcal{A}$ denote random variables on time $t$, and $s, s' \in \mathcal{S}$ and $a, b \in \mathcal{A}$ denote state and action instances. $\mathcal{P}(s, a, s') = \mathbb{P}(S_{t+1} = s'|S_t = s, A_t = a)$ defines the transition dynamics and $r(s, a) \in \mathbb{R}$ defines the expected immediate reward function. A (soft-)greedy policy $\pi^*(a|s, Q)$ is a (stochastic) mapping from states to actions and is based on the value function $Q$. A parameterized policy $\pi(a|s, \theta)$ is a stochastic mapping from states to actions and is based on the parameter vector $\theta$. Note that we use $\pi^*$ to denote a (soft-)greedy policy, not an optimal policy. The action value functions $Q(s, a)$ and $A(s, a)$ are estimators of the $\gamma$-discounted cumulative reward $\sum_t \gamma^t \mathbb{E}[r(S_t, A_t)|S_0 = s, A_0 = a, \pi]$ that follows some $(s, a)$ under some $\pi$. The state value function $V$ is an estimator of such cumulative reward that follows some $s$.

In policy iteration, the current policy is fully evaluated, after which a policy improvement step is taken based on this evaluation. In optimistic policy iteration, policy improvement is based on an incomplete evaluation. In value iteration, just a one-step lookahead improvement is made at a time.

In greedy value function reinforcement learning (e.g., [2, 3]), the current policy on iteration $k$ is usually implicit and is greedy (and thus deterministic) with respect to the value function $Q_{k-1}$ of the previous policy:

$$\pi^*(a|s, Q_{k-1}) = \begin{cases} 1 & \text{if } a = \arg\max_b Q_{k-1}(s, b) \\ 0 & \text{otherwise.} \end{cases} \tag{1}$$

Improvement is obtained by estimating a new value function $Q_k$ for this policy, after which the process repeats. Soft-greedy iteration is obtained by slightly softening $\pi^*$ in some way so that

$\pi^*(a|s, Q_{k-1}) > 0, \forall a, s$, the Gibbs soft-greedy policy class with a temperature $\tau$ (Boltzmann exploration) being a common choice:

$$\pi^*(a|s, Q_{k-1}) \propto e^{Q_{k-1}(s,a)/\tau} \ . \tag{2}$$

We note that (1) becomes approximated by (2) arbitrarily closely as $\tau \to 0$ and that this corresponds to scaling the action values toward infinity.

A common choice for approximating $Q$ is to obtain a least-squares fit using a linear-in-parameters approximator $\tilde{Q}$ with the feature basis $\phi^*$:

$$\tilde{Q}_k(s, a, w_k) = w_k^\top \phi^*(s, a) \approx Q_k(s, a) \ . \tag{3}$$

For the soft-greedy case, one option is to use an approximator that will obtain an approximation of an advantage function (see [9]):[1]

$$\tilde{A}_k(s, a, w_k) = w_k^\top \left( \phi^*(s, a) - \sum_b \pi^*(b|s, \tilde{A}_{k-1}) \phi^*(s, b) \right) \approx A_k(s, a) \ . \tag{4}$$

Convergence properties depend on how the estimation is performed and on the function approximator class with which $Q$ is being approximated. For greedy approximate policy iteration in the general case, policy convergence is guaranteed only up to bounded sustained oscillation [2]. Optimistic variants can permit asymptotic convergence in parameters, although the corresponding policy can manifest sustained oscillation even then [8, 2, 7]. For the case of greedy approximate value iteration, a line of research has provided solid (although restrictive) conditions for the approximator class for having asymptotic parameter convergence (reviewed in, e.g., [3]), whereas the question of policy convergence in these cases has been left quite open. In the rest of the paper, our focus will be on non-optimistic approximate policy iteration.

In policy gradient reinforcement learning (e.g., [9, 6, 4, 5]), the current policy on iteration $k$ is explicitly represented using some differentiable stochastic policy class $\pi(\theta)$, the Gibbs policy with some basis $\phi$ being a common choice:

$$\pi(a|s, \theta) \propto e^{\theta^\top \phi(s,a)} \ . \tag{5}$$

Improvement is obtained via stochastic gradient ascent: $\theta_{k+1} = \theta_k + \alpha_k \partial J(\theta_k)/\partial \theta$. In actor-critic (value-based policy gradient) methods that implement a policy gradient based approximate policy iteration scheme, the so-called 'compatibility condition' is fulfilled if the value function is approximated using (4) with $\phi^* = \phi$ and $\pi(\theta_k)$ in place of $\pi^*(\tilde{A}_{k-1})$ (e.g., [9]). In this case, the value function parameter vector $w$ becomes the natural gradient estimate $\eta$ for the policy $\pi(a|s, \theta)$, leading to the natural actor-critic algorithm [13, 4]:

$$\eta = w \ . \tag{6}$$

Here, convergence w.p.1 to a local optimum is established for Monte Carlo evaluation under standard assumptions (properly diminishing step-sizes and ergodicity of the sampling process, roughly speaking) [9, 6]. Convergence into bounded suboptimality is obtained under temporal difference evaluation [6, 5].

## 3 The policy oscillation phenomenon

It is well known that greedy policy iteration can be non-convergent under approximations. The widely used projected equation approach can manifest convergence behavior that is complex and not well understood, including bounded but potentially severe sustained policy oscillations [7, 8, 18]. Similar consequences arise in the context of partial observability for approximate or incomplete state estimation (e.g., [19, 20, 21]).

It is important to remember that sustained policy oscillation can take place even under (asymptotic) value function convergence (e.g., [7, 8]). Policy convergence can be established under various restrictions. Continuously soft-greedy action selection (which is essentially a step toward the policy

gradient approach) has been found to have a beneficial effect in cases of both value function approximation and partial observability [22]. A notable approach is introduced in [7] wherein it is also shown that the suboptimality bound for a converging policy sequence is much better. Interestingly, for the special case of Monte Carlo estimation of action values, it is also possible to establish convergence by solely modifying the exploration scheme, which is known as consistent exploration [23] or MCESP [24].

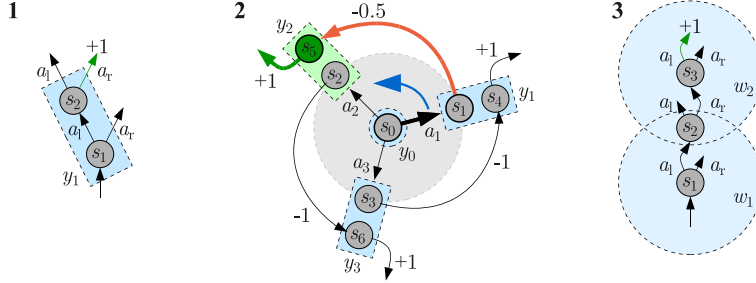

Figure 1: Oscillatory examples. Boxes marked with $y_k$ denote observations (aggregated states). Circles marked with $w_k$ illustrate receptive fields of the basis functions. Only non-zero rewards are shown. Start states: $s_1$ (1.1), $s_0$ (1.2), and $s_1$ (1.3). Arrows leading out indicate termination.

The setting likely to be the simplest possible in which oscillation occurs even with Monte Carlo policy evaluation is depicted in Figure 1.1 (adapted from [21]). The actions $a_l$ and $a_r$ are available in the decision states $s_1$ and $s_2$. Both states are observed as $y_1$. The only reward is obtained with the decision sequence $(s_1, a_l; s_2, a_r)$. Greedy value function methods that operate without state estimation will oscillate between the policies $\pi(y_1) = a_l$ and $\pi(y_1) = a_r$, excluding the exceptions mentioned above. This example can also be used to illustrate how local optima can arise in the presence of approximations by changing the sign of the reward that follows $(s_2, a_r)$ (see [20]). Figure 1.2 (adapted from [25]) shows a more complex case in which a deterministic optimal solution *is* attainable. The actions $a_{[1,3]}$ are available in the only decision state $s_0$, which is observed as $y_0$. Oscillation will occur when using temporal difference evaluation but not with Monte Carlo evaluation. These two POMDP examples are trivially equivalent to value function approximation using hard aggregation. Figure 1.3 (a novel example inspired by the classical XOR problem) shows how similar counterexamples can be constructed also for the case of softer value function approximation. The action values are approximated with $\tilde{Q}(s_1, a_l) = w_{1,l}$, $\tilde{Q}(s_2, a_l) = 0.5w_{1,l} + 0.5w_{2,l}$, $\tilde{Q}(s_3, a_l) = w_{2,l}$, $\tilde{Q}(s_1, a_r) = w_{1,r}$, $\tilde{Q}(s_2, a_r) = 0.5w_{1,r} + 0.5w_{2,r}$, and $\tilde{Q}(s_3, a_r) = w_{2,r}$. The only reward is obtained with the decision sequence $(s_1, a_l; s_2, a_r; s_3, a_l)$. Oscillation will occur even with Monte Carlo evaluation. For other examples, see e.g. [8, 19].

A detailed description of the oscillation phenomenon can be found in [8, §6.4] (see also [12, 7]), where it is described in terms of cyclic sequences in the so-called greedy partition of the value function parameter space. Although this line of research has provided a concise description of the phenomenon, it has not fully answered the question of *why* approximations can introduce such cyclic sequences in greedy policy iteration and why strong convergence guarantees exist for the policy gradient based counterpart of this methodology. We will proceed by taking a different view by casting a considerable subset of the former approach as a special case of the latter.

## 4   Approximations and attractive stochastic policies

In this section, we briefly and informally examine how policy oscillation arises in the examples in Section 3. In all cases, oscillation is caused by the presence of an attractive stochastic policy, these attractors being induced by approximations. In the case of partial observability without proper state estimation (Figure 1.1), the policy class is incapable of representing differing action distributions for the same observation with differing histories. This makes the optimal sequence $(y_1, a_l; y_1, a_r)$ inexpressible for deterministic policies, whereas a stochastic policy can still emit it every now and then by chance. In Figure 1.3, the same situation is arrived at due to the insufficient capacity of the approximator: the specified value function approximator cannot express such value estimates that

would lead to an implicit greedy policy that attains the optimal sequence $(s_1, a_l; s_2, a_r; s_3, a_l)$. Generally speaking, in these cases, oscillation follows from a mismatch between the main policy class and the exploration policy class: stochastic exploration can occasionally reach the reward, but the deterministic main policy is incapable of exploiting this opportunity. The opportunity nevertheless keeps appearing in the value function, leading to repeated failing attempts to exploit it. Consistent exploration avoids the problem by limiting exploration to only expressible sequences.

Temporal difference evaluation effectively solves for an implicit Markov model [26], i.e., it gains variance reduction in policy evaluation by making the Markov assumption. When this assumption fails, the value function shows non-existent improvement opportunities. In Figure 1.2, an incorrect Markov assumption leads to a TD solution that corresponds to a model in which, e.g., the actually impossible sequence $(y_0, a_2, r = 0; y_2, -, r = +1; \text{end})$ becomes possible and attractive. Generally speaking, oscillation results in this case from perceived but non-existent improvement opportunities that vanish once an attempt is made to exploit them. This vanishing is caused by changes in the sampling distribution that leads to a different implicit Markov model and, consequently, to a different fixed point (see [27, 18]).

In summary, stochastic policies can become attractive due to deterministically unreachable or completely non-existing improvement opportunities that appear in the value function. In all cases, the class of stochastic policies allows gradually increasing the attempt of exploitation of such an opportunity until it is either optimally exploited or it has vanished enough so as to have no advantage over alternatives, at which point a stochastic equilibrium is reached.

## 5  Policy oscillation as sustained overshooting

In this section, we focus more carefully on how attractive stochastic policies lead to sustained policy oscillation when viewed within the policy gradient framework. We begin by looking at a natural actor-critic algorithm that uses the Gibbs policy class (5). We iterate by fully estimating $\tilde{A}_k$ in (4) for the current policy $\pi(\theta_k)$, as shown in [4], and then a gradient update is performed using (6):

$$\theta_{k+1} = \theta_k + \alpha\eta_k . \tag{7}$$

Now let us consider some policy $\pi(\theta_k)$ from such a policy sequence generated by (7) and denote the corresponding value function estimate by $\tilde{A}_k$ and the natural gradient estimate by $\eta_k$. It is shown in [13] that taking a very long step in the direction of the natural gradient $\eta_k$ will approach in the limit a greedy update (1) for the value function $\tilde{A}_k$:

$$\pi(a|s, \theta_k + \alpha\eta_k) \overset{\lim}{=} \pi^*(a|s, \tilde{A}_k) , \quad \alpha \to \infty, \ \theta_k \nrightarrow \infty, \ \eta \neq 0, \ \forall s, a . \tag{8}$$

The resulting policy will have the form

$$\pi(a|s, \theta_k + \alpha\eta_k) \propto e^{\theta_k^\top \phi(s,a) + \alpha\eta_k^\top \phi(s,a)} . \tag{9}$$

The proof in [13] is based on the term $\alpha\eta_k^\top \phi(s, a)$ dominating the sum when $\alpha \to \infty$. Thus, this type of a greedy update is a special case of a natural gradient update in which the step-size approaches infinity.

However, the requirement that $\theta_k \nrightarrow \infty$ will hold only during the first step using a constant $\alpha \to \infty$, assuming a bounded initial $\theta$. Thus, natural gradient based policy iteration using such a very large but constant step-size does *not* approach greedy value function based policy iteration after the first such iteration. Little is needed, however, to make the equality apply in the case of full policy iteration. The cleanest way, in theory, is to use a steeply *increasing* step-size schedule.

**Theorem 1.** *Let $\pi(\theta_k)$ denote the kth policy obtained from (7) using the step-sizes $\alpha_{[0,k-1]}$ and natural gradients $\eta_{[0,k-1]}$. Let $\pi^*(w_k)$ denote the kth policy obtained from (1) with infinitely small added softness and using a value function (4), with $\phi^* = \phi$ and $\tilde{A}(w_0)$ being evaluated for $\pi(\theta_0)$. Assume $\theta_0$ to be bounded from infinity. Assume all $\eta_k$ to be bounded from zero and infinity. If $\alpha_0 \to \infty$ and $\alpha_k/\alpha_{k-1} \to \infty, \ \forall k > 0$, then $\pi(\theta_{k+1}) =^{\lim} \pi^*(w_k)$.*

*Proof.* The equivalence after the first iteration is proven in [13] with the requirement that the sum in (9) is dominated by the last term $\alpha_0\eta_0^\top \phi(s, a)$. For $\alpha_0 \to \infty$, this holds if $\theta_0$ is bounded and

$\eta_0 \neq 0$. By writing the parameter vector after the second iteration as $\theta_2 = \theta_0 + \alpha_0 \eta_0 + \alpha_1 \eta_1$, the sum becomes

$$\theta_0^\top \phi(s,a) + \alpha_0 \eta_0^\top \phi(s,a) + \alpha_1 \eta_1^\top \phi(s,a) . \tag{10}$$

The requirement for the result in [13] to still apply is that the last term keeps dominating the sum. Assuming $\theta_0 \not\to \infty$, $\eta_0 \not\to \infty$, and $\eta_1 \neq 0$, then this condition is maintained if $\alpha_1 \to \infty$ and $\alpha_1/\alpha_0 \to \infty$. That is, the step-size in the second iteration needs to approach infinity much faster than the step-size in the first iteration. The rest follows by induction. $\qquad\square$

However, once the first update is performed using such a very large step-size, it is no longer possible to revert back to more conventional step-size schedules: once $\theta$ has become very large, any update performed using a much smaller $\alpha$ will have virtually no effect on the policy. In the following, a more practical alternative is discussed that both avoids the related numerical issues and that allows gradual interpolation back toward conventional policy gradient iteration. It also makes it easier to illustrate the resulting process, which we will do shortly. However, a slight modification to the natural actor-critic algorithm is required.

More precisely, we constrain the magnitude of $\theta$ by enforcing $\|\theta\| \leq c$ after each update, where $\|\theta\|$ is some measure of the magnitude of $\theta$ and $c$ is some positive constant. Here the update equation (7) is replaced by:

$$\theta_{k+1} = \begin{cases} \theta_k + \alpha\eta_k & \text{if } \|\theta_k + \alpha\eta_k\| \leq c \\ \tau_c(\theta_k + \alpha\eta_k) & \text{otherwise,} \end{cases} \tag{11}$$

where $\tau_c = c/\|\theta_k + \alpha\eta_k\|$.

**Theorem 2.** *Let $\pi(\theta_k)$ and $\pi^*(w_k)$ be as previously, except that (7) is replaced with (11). Let $\tilde{A}(w_0)$ to be evaluated for $\pi(\theta_0)$. Assume $\theta_0 \not\to \infty$ and all $\eta_k \neq 0$. If $c \to \infty$ and $\alpha/c \to \infty$, then $\pi(\theta_{k+1}) \overset{\lim}{=} \pi^*(w_k)$.*

*Proof.* The proof in [13] for a single iteration of the unmodified algorithm requires that the last term of the sum in (9) dominates. This holds if $\alpha/\|\theta_k\| \to \infty$ and $\eta_k \neq 0$. This is ensured during the first iteration by having $\theta_0 \not\to \infty$. After the $k$th iteration, $\|\theta_k\| \leq c$ due to the constraint in (11), and the last term will dominate as long as $\alpha/c \to \infty$ and $\eta_k \neq 0$.

The constraint affects the policy $\pi(\theta_{k+1})$ only when $\|\theta_k + \alpha\eta_k\| > c$, in which case the magnitude of the parameter vector is scaled down with a factor $\tau_c$ so that it becomes equal to $c$. This has a diminishing effect on the resulting policy as $c \to \infty$ because the Gibbs distribution becomes increasingly insensitive to scaling of the parameter vector when its magnitude approaches infinity:

$$\pi(\tau_c\theta) \overset{\lim}{=} \pi(\theta) , \quad \forall \tau_c, \theta \text{ such that } \|\tau_c\theta\| \to \infty, \ \|\theta\| \to \infty . \tag{12}$$

$\qquad\square$

With a constant $\alpha \to \infty$ and finite $c$, the resulting constrained natural actor-critic algorithm (CNAC) is analogous to soft-greedy iteration in which on-policy Boltzmann exploration with temperature $\tau = 1/c$ is used: constraining the magnitude of $\theta$ will effectively ensure some minimum level of stochasticity in the corresponding policy (there is a mismatch between the algorithms even for $\tau = 1/c$ whenever $\|\eta\| \neq 1$). If the soft-greedy method uses (4) for policy evaluation, then exact equivalence in the limit is obtained when $c \to \infty$ while maintaining $\alpha/c \to \infty$. Lowering $\alpha$ interpolates toward a conventional natural gradient method. These considerations apply also for (3) in place of (4) in the soft-greedy method if the indices of the maximizing actions become estimated equally in both cases: $\arg\max_a \tilde{A}(s,a,w_A) = \arg\max_b \tilde{Q}(s,b,w_Q)$, $\forall s$.

Greedy policy iteration searches in the space of deterministic policies. As noted, the sequence of greedy policies that is generated by such a process can be approximated arbitrarily closely with the Gibbs policy class (2) with $\tau \to 0$. For this class, the parameters of all deterministic policies lie at infinity in different directions in the parameter space, whereas stochastic policies are obtained with finite parameter values (except for vanishingly narrow directions along diagonals). From this point of view, the search is conducted on the surface of an $\infty$-radius sphere: each iteration performs a jump from infinity in one direction to infinity in some other direction. Based on Theorems 1 and 2, we observe that the policy sequence that results from these jumps can be approximated arbitrarily closely with a natural actor-critic method using very large step-sizes.

The soundness of such a process obviously requires some special structure for the gradient landscape. In informal terms, what suffices is that the performance landscape has a monotonically increasing profile up to infinity in the direction of a gradient that is estimated at any point. This condition is established if all attractors in the parameter space reside at infinity and if the gradient field is not, loosely speaking, too 'curled'. Although we ignore the latter requirement, we note that the former requirement is satisfied when only deterministic attractors exist in the Gibbs policy space. This condition holds when the problem is fully Markovian and the value function is represented exactly, which leads to the standard result for MDPs stating that there always exists a deterministic policy that is globally optimal, that there are no locally optimal policies and that any potential stochastic optimal policies are not attractive (e.g., [1, §A.2]). However, when these conditions do not hold and there is an attractor in the policy space that corresponds to a stochastic policy, there is a finite attractor in the parameter space that resides *inside* the $\infty$-radius sphere.

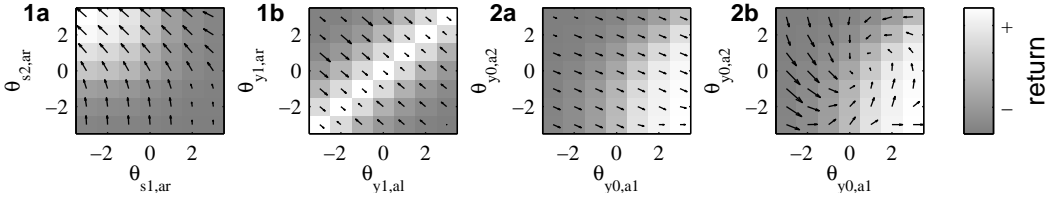

Figure 2: Performance landscapes and estimated gradient fields for the examples in Figure 1.

The required special structure is clearly visible in Figure 2.1a, in which the performance landscape and the gradient field are shown for the fully observable (and thus Markovian) counterpart of the problem from Figure 1.1. This structure can be seen also in Figure 2.2a, in which the problem from Figure 1.2 is evaluated using Monte Carlo evaluation. The redundant parameters $\theta_{s_1,a_l}$ and $\theta_{s_2,a_l}$ in the former and $\theta_{y_0,a_3}$ in the latter were fixed to zero. In these cases, movement in the direction of the natural gradient keeps improving performance up to infinity, i.e., there are no finite optima in the way. This structure is clearly lost in Figure 2.1b, which shows the evaluation for the non-Markovian problem from Figure 1.1. The same holds for the temporal differences based gradient field for the problem from Figure 1.2 that is shown in Figure 2.2b. In essence, the sustained policy oscillation that results from using very large step-sizes or greedy updates in the latter two cases (2.1b and 2.2b) is caused by sustained overshooting over the finite attractor in the policy parameter space.

Another implication of the equivalence between very long natural gradient updates and greedy updates is that, contrary to what is sometimes suggested in the literature, the natural actor-critic approach has an inherent capability for a speed that is comparable to parametric greedy approaches with linear-in-parameters value function approximation. This is because whatever initial improvement speed can be achieved with the latter due to greedy updates, the same speed can be also achieved with the former using the same basis together with very long steps and constraining. This effectively corresponds to an attempt to exploit whatever remains of the special structure of a Markovian problem, making the use of a very large $\alpha$ in constrained policy improvement analogous to using a small $\lambda$ in policy evaluation. Constraining $\|\theta\|$ also enables interpolating back toward conventional natural policy gradient learning (in addition to offering a crude way of maintaining explorativity): in cases of partial Markovianity, very long (near-infinite) natural gradient steps can be used to quickly find the rough direction of the strongest attractors, after which gradually decreasing the step-size allows an ascent toward some finite attractor.

## 6 Empirical results

In this section, we apply several variants of the natural actor-critic algorithm and some greedy policy iteration algorithms to the Tetris problem using the standard features from [12]. For policy improvement, we use the original natural actor-critic (NAC) from [4], a constrained one (CNAC) that uses (11) and a very large $\alpha$, and a soft-greedy policy iteration algorithm (SGPI) that uses (2). For policy evaluation, we use LSPE [28] and an SVD-based batch solver (pinv). The $B$ matrix in LSPE was initialized to $0.5I$ and the policy was updated after every 100th episode. We used the advantage estimator from (4) unless otherwise stated. We used a simple initial policy ($\theta^{\mathrm{maxh}} = \theta^{\mathrm{holes}} = -3$) that scores around 20 points.

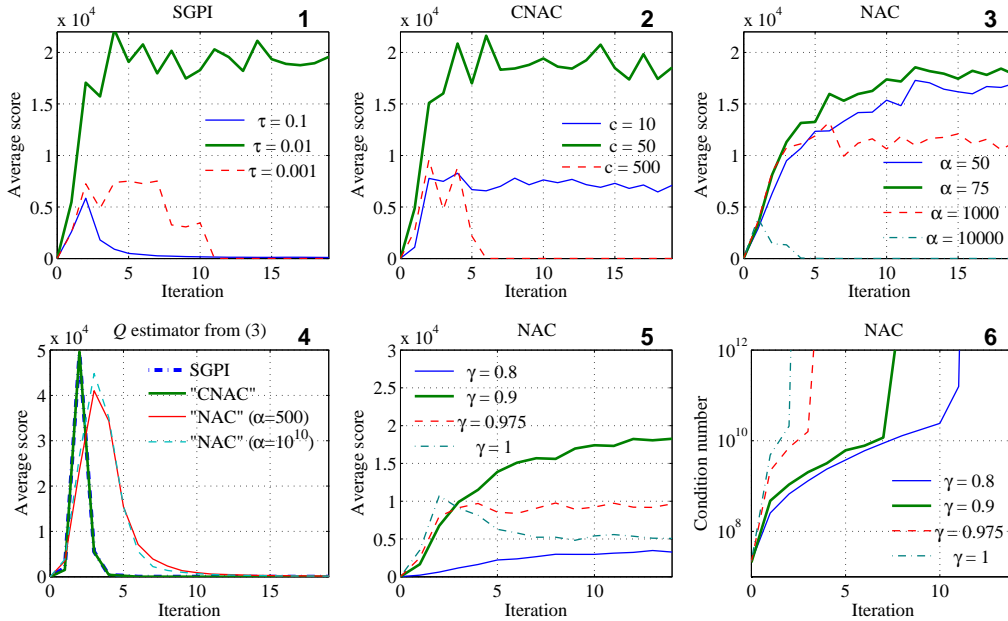

Figure 3: Empirical results for the Tetris problem. See the text for details.

Figure 3.1 shows that with soft-greedy policy iteration (SGPI), it is in fact possible to avoid policy degradation by using a suitable amount of softening. Results for constrained natural actor-critic (CNAC) with $\alpha = 10^{10}$ are shown in Figure 3.2. The algorithm can indeed emulate greedy updates (SGPI) and the associated policy degradation. Unconstrained natural actor-critic (NAC), shown in Figure 3.3, failed to match the behavior and speed of SGPI and CNAC with any step-size (only selected step-sizes are shown). Results for all algorithms when using the $Q$ estimator in (3) are shown in Figure 3.4 (technically, CNAC and NAC are not using a natural gradient now). SGPI and CNAC match perfectly while reaching transiently a level around 50,000 points in just 2 iterations.

We did observe a presence of oscillation-predisposing structure during several runs. There were optima at finite parameter values along several consecutively estimated gradient directions, but these optima did not usually form closed basins of attraction in the full parameter space. At such points, the performance landscape was reminiscent of what was illustrated in Figure 2.1b, except that there was a tendency for a slope toward an open end of the valley (ridge) at finite distance. As a result, oscillations were mainly transient with suitably chosen learning parameter values.

However, a commonality among all the algorithms was that the relevant matrices became quickly highly ill-conditioned. This was the case especially when using (4), with which condition numbers were typically above $10^9$ upon convergence/stagnation. Figures 3.5 and 3.6 show performance levels and typical condition numbers for NAC with different discount factors. It can be seen that the inferior results obtained with a too high $\gamma$ (cf. [14, 12]) are associated with serious ill-conditioning.

In contrast to typical approximate dynamic programming methods, the cross-entropy method involves numerically more stable computations and, moreover, the computations are based on information from a *distribution* of policies. Currently, we expect that the policy oscillation or chattering phenomenon is not the main cause for neither policy degradation nor stagnation in this problem. Instead, it seems that, for both greedy and gradient approaches, the explanation is related to numerical instabilities that stem possibly both from the involved estimators and from insufficient exploration.

### Acknowledgments

We thank Lucian Buşoniu and Dimitri Bertsekas for valuable discussions. This work has been financially supported by the Academy of Finland through the Centre of Excellence Programme.

## Footnotes

[1]The approach in [4] is needed to permit temporal difference evaluation in this case.

## References

[1] C. Szepesvári. *Algorithms for reinforcement learning*. Morgan & Claypool Publishers, 2010.

[2] D. P. Bertsekas. *Dynamic Programming and Optimal Control*. Athena Scientific, 2005.

[3] L. Buşoniu, R. Babuška, B. De Schutter, and D. Ernst. *Reinforcement learning and dynamic programming using function approximators*. CRC Press, 2010.

[4] J. Peters and S. Schaal. Natural actor-critic. *Neurocomputing*, 71(7-9):1180–1190, 2008.

[5] S. Bhatnagar, R. S. Sutton, M. Ghavamzadeh, and M. Lee. Natural actor-critic algorithms. *Automatica*, 45(11):2471–2482, 2009.

[6] V. R. Konda and J. N. Tsitsiklis. On actor-critic algorithms. *SIAM Journal on Control and Optimization*, 42(4):1143–1166, 2004.

[7] D. P. Bertsekas. Approximate policy iteration: A survey and some new methods. Technical report, Massachusetts Institute of Technology, Cambridge, US, 2010.

[8] D. P. Bertsekas and J. N. Tsitsiklis. *Neuro-dynamic programming*. Athena Scientific, 1996.

[9] R. S. Sutton, D. Mcallester, S. Singh, and Y. Mansour. Policy gradient methods for reinforcement learning with function approximation. In *Advances in Neural Information Processing Systems*, 2000.

[10] C. Thiery and B. Scherrer. Building Controllers for Tetris. *ICGA Journal*, 32(1):3–11, 2009.

[11] I. Szita and A. Lörincz. Learning Tetris using the noisy cross-entropy method. *Neural Computation*, 18(12):2936–2941, 2006.

[12] D. P. Bertsekas and S. Ioffe. Temporal differences-based policy iteration and applications in neuro-dynamic programming. Technical report, Massachusetts Institute of Technology, Cambridge, US, 1996.

[13] S. M. Kakade. A natural policy gradient. In *Advances in Neural Information Processing Systems*, 2002.

[14] M. Petrik and B. Scherrer. Biasing approximate dynamic programming with a lower discount factor. In *Advances in Neural Information Processing Systems*, 2008.

[15] C. Thiery and B. Scherrer. Improvements on learning Tetris with cross entropy. *ICGA Journal*, 32(1):23–33, 2009.

[16] V. Farias and B. Roy. Tetris: A study of randomized constraint sampling. In *Probabilistic and Randomized Methods for Design Under Uncertainty*, pages 189–201. Springer, 2006.

[17] V. Desai, V. Farias, and C. Moallemi. A smoothed approximate linear program. In *Advances in Neural Information Processing Systems*, 2009.

[18] G. J. Gordon. Reinforcement learning with function approximation converges to a region. In *Advances in Neural Information Processing Systems*, 2001.

[19] S. P. Singh, T. Jaakkola, and M. I. Jordan. Learning without state-estimation in partially observable markovian decision processes. In *Proceedings of the Eleventh International Conference on Machine Learning*, volume 31, page 37, 1994.

[20] M. D. Pendrith and M. J. McGarity. An analysis of direct reinforcement learning in non-markovian domains. In *Proceedings of the Fifteenth International Conference on Machine Learning*, 1998.

[21] T. J. Perkins. Action value based reinforcement learning for POMDPs. Technical report, University of Massachusetts, Amherst, MA, USA, 2001.

[22] T. J. Perkins and D. Precup. A convergent form of approximate policy iteration. In *Advances in Neural Information Processing Systems*, 2003.

[23] P. A. Crook and G. Hayes. Consistent exploration improves convergence of reinforcement learning on POMDPs. In *AAMAS 2007 Workshop on Adaptive and Learning Agents*, 2007.

[24] T. J. Perkins. Reinforcement learning for POMDPs based on action values and stochastic optimization. In *Proceedings of the Eighteenth National Conference on Artificial Intelligence*, pages 199–204. American Association for Artificial Intelligence, 2002.

[25] G. J. Gordon. Chattering in SARSA($\lambda$). Technical report, Carnegie Mellon University, Pittsburgh, PA, USA, 1996.

[26] R. Parr, L. Li, G. Taylor, C. Painter-Wakefield, and M. L. Littman. An analysis of linear models, linear value-function approximation, and feature selection for reinforcement learning. In *Proceedings of the 25th International Conference on Machine learning*, pages 752–759. ACM, 2008.

[27] D. P. Bertsekas and H. Yu. Q-learning and enhanced policy iteration in discounted dynamic programming. In *Decision and Control (CDC), 2010 49th IEEE Conference on*, pages 1409–1416. IEEE, 2010.

[28] A. Nedić and D. P. Bertsekas. Least squares policy evaluation algorithms with linear function approximation. *Discrete Event Dynamic Systems: Theory and Applications*, 13(1–2):79–110, 2003.

